# Simplified Rules and Theoretical Analysis for Information Bottleneck Optimization and PCA with Spiking Neurons

**Lars Buesing, Wolfgang Maass**
Institute for Theoretical Computer Science
Graz University of Technology
A-8010 Graz, Austria
{lars,maass}@igi.tu-graz.at

## Abstract

We show that under suitable assumptions (primarily linearization) a simple and perspicuous online learning rule for Information Bottleneck optimization with spiking neurons can be derived. This rule performs on common benchmark tasks as well as a rather complex rule that has previously been proposed [1]. Furthermore, the transparency of this new learning rule makes a theoretical analysis of its convergence properties feasible. A variation of this learning rule (with sign changes) provides a theoretically founded method for performing Principal Component Analysis (PCA) with spiking neurons. By applying this rule to an ensemble of neurons, different principal components of the input can be extracted. In addition, it is possible to preferentially extract those principal components from incoming signals $X$ that are related or are not related to some additional target signal $Y_T$. In a biological interpretation, this target signal $Y_T$ (also called relevance variable) could represent proprioceptive feedback, input from other sensory modalities, or top-down signals.

## 1  Introduction

The Information Bottleneck (IB) approach [2] allows the investigation of learning algorithms for unsupervised and semi-supervised learning on the basis of clear optimality principles from information theory. Two types of time-varying inputs $X$ and $Y_T$ are considered. The learning goal is to learn a transformation from $X$ into another signal $Y$ that extracts only those components from $X$ that are related to the relevance signal $Y_T$. In a more global biological interpretation $X$ might represent for example some sensory input, and $Y$ the output of the first processing stage for $X$ in the cortex. In this article $Y$ will simply be the spike output of a neuron that receives the spike trains $X$ as inputs. The starting point for our analysis is the first learning rule for IB optimization in for this setup, which has recently been proposed in [1], [3]. Unfortunately, this learning rule is complicated, restricted to discrete time and no theoretical analysis of its behavior is feasible. Any online learning rule for IB optimization has to make a number of simplifying assumptions, since true IB optimization can only be carried out in an offline setting. We show here, that with a slightly different set of assumptions than those made in [1] and [3], one arrives at a drastically simpler and intuitively perspicuous online learning rule for IB optimization with spiking neurons. The learning rule in [1] was derived by maximizing the objective function[1] $L_0$:

$$L_0 = -I(X, Y) + \beta I(Y, Y_T) - \gamma D_{KL}(P(Y) \| P(\tilde{Y})), \tag{1}$$

where $I(.,.)$ denotes the mutual information between its arguments and $\beta$ is a positive trade-off factor. The target signal $Y_T$ was assumed to be given by a spike train. The learning rule from [1] (see [3] for a detailed interpretation) is quite involved and requires numerous auxiliary definitions (hence we cannot repeat it in this abstract). Furthermore, it can only be formulated in discrete time (steps size $\Delta t$) for reasons we want to outline briefly: In the limit $\Delta t \to 0$ the essential contribution to the learning rule, which stems from maximizing the mutual information $I(Y, Y_T)$ between output and target signal, vanishes. This difficulty is rooted in a rather technical assumption, made in appendix A.4 in [3], concerning the expectation value $\overline{\overline{\rho}}^k$ at time step $k$ of the neural firing probability $\rho$, given the information about the postsynaptic spikes and the target signal spikes up to the preceding time step $k-1$ (see our detailed discussion in [4])[2]. The restriction to discrete time prevents the application of powerful analytical methods like the Fokker-Planck equation, which requires continuous time, for analyzing the dynamics of the learning rule.

In section 2 of this paper, we propose a much simpler learning rule for IB optimization with spiking neurons, which can also be formulated in continuous time. In contrast to [3], we approximate the critical term $\overline{\overline{\rho}}^k$ with a linear estimator, under the assumption that $X$ and $Y_T$ are positively correlated. Further simplifications in comparison to [3] are achieved by considering a simpler neuron model (the linear Poisson neuron, see [5]). However we show through computer simulation in [4] that the resulting simple learning rule performs equally well for the more complex neuron model with refractoriness from [1] - [5]. The learning rule presented here can be analyzed by the means of the drift function of the corresponding Fokker-Planck equation. The theoretical results are outlined in section 3, followed by the consideration of a concrete IB optimization task in section 4. A link between the presented learning rule and Principal Component Analysis (PCA) is established in section 5. A more detailed comparison of the learning rule presented here and the one of [3] as well as results of extensive computer tests on common benchmark tasks can be found in [4].

## 2 Neuron model and learning rule for IB optimization

We consider a linear Poisson neuron with $N$ synapses of weights $w = (w_1, \ldots, w_N)$. It is driven by the input $X$, consisting of $N$ spike trains $X_j(t) = \sum_i \delta(t - t_j^i)$, $j \in \{1, \ldots, N\}$, where $t_j^i$ denotes the time of the i'th spike at synapse $j$. The membrane potential $u(t)$ of the neuron at time $t$ is given by the weighted sum of the presynaptic activities $\nu(t) = (\nu_1(t), \ldots, \nu_N(t))$:

$$
\begin{align}
u(t) &= \sum_{j=1}^{N} w_j \nu_j(t) \tag{2} \\
\nu_j(t) &= \int_{-\infty}^{t} \epsilon(t-s) X_j(s) ds.
\end{align}
$$

The kernel $\epsilon(.)$ models the EPSP of a single spike (in simulations $\epsilon(t)$ was chosen to be a decaying exponential with a time constant of $\tau_m = 10\,\mathrm{ms}$). The postsynaptic neuron spikes at time $t$ with the probability density $g(t)$:

$$
g(t) = \frac{u(t)}{u_0},
$$

with $u_0$ being a normalization constant. The postsynaptic spike train is denoted as $Y(t) = \sum_i \delta(t - t_f^i)$, with the firing times $t_f^i$.

We now consider the IB task described in general in [2], which consists of maximizing the objective function $L_{\mathrm{IB}}$, in the context of spiking neurons. As in [6], we introduce a further term $L_3$ into the the objective function that reflects the higher metabolic costs for the neuron to maintain strong synapses, a natural, simple choice being $L_3 = -\lambda \sum w_j^2$. Thus the complete objective function $L$ to maximize is:

$$
L = L_{\mathrm{IB}} + L_3 = -I(X, Y) + \beta I(Y_T, Y) - \lambda \sum_{j=1}^{N} w_j^2. \tag{3}
$$

The objective function $L$ differs slightly from $L_0$ given in (1), which was optimized in [3]; this change turned out to be advantageous for the PCA learning rule given in section 5, without significantly changing the characteristics of the IB learning rule.

The online learning rule governing the change of the weights $w_j(t)$ at time $t$ is obtained by a gradient ascent of the objective function $L$:

$$\frac{d}{dt} w_j(t) = \alpha \frac{\partial L}{\partial w_j}.$$

For small learning rates $\alpha$ and under the assumption that the presynaptic input $X$ and the target signal $Y_T$ are stationary processes, the following learning rule can be derived:

$$\frac{d}{dt} w_j(t) = \alpha \frac{Y(t)\nu_j(t)}{u(t)\overline{u}(t)} \left( -(u(t) - \overline{u}(t)) + \beta \left( F[Y_T](t) - \overline{F[Y_T]}(t) \right) \right) - \alpha \lambda w_j(t), \quad (4)$$

where the operator $\overline{(.)}$ denotes the low-pass filter with a time constant $\tau_C$ (in simulations $\tau_C = 3s$), i. e. for a function $f$:

$$\overline{f}(t) = \frac{1}{\tau_C} \int_{-\infty}^{t} \exp\left( -\frac{t-s}{\tau_C} \right) f(s)ds. \quad (5)$$

The operator $F[Y_T](t)$ appearing in (4) is equal to the expectation value of the membrane potential $\langle u(t) \rangle_{X|Y_T} = E[u(t)|Y_T]$, given the observations $(Y_T(\tau)|\tau \in \mathbb{R})$ of the relevance signal; $F$ is thus closely linked to estimation and filtering theory. For a known joint distribution of the processes $X$ and $Y_T$, the operator $F$ could in principal be calculated exactly, but it is not clear how this quantity can be estimated in an online process; thus we look for a simple approximation to $F$. Under the above assumptions, $F$ is time invariant and can be approximated by a Volterra series (for details see [4]):

$$\langle u(t) \rangle_{X|Y_T} = F[Y_T](t) = \sum_{n=0}^{\infty} \int_{\mathbb{R}} \cdots \int_{\mathbb{R}} \kappa_n(t - t_1, \ldots, t - t_n) \prod_{i=1}^{n} Y_T(t_i) dt_i. \quad (6)$$

In this article, we concentrate on the situation, where F can be well approximated by its linearization $F_1[Y_T](t)$, corresponding to a linear estimator of $\langle u(t) \rangle_{X|Y_T}$. For $F_1[Y_T](t)$ we make the following ansatz:

$$F[Y_T](t) \approx F_1[Y_T](t) = c \cdot u_T(t) = c \int_{\mathbb{R}} \kappa_1(t - t_1) Y_T(t_1) dt_1. \quad (7)$$

According to (7), $F$ is approximated by a convolution $u_T(t)$ of the relevance signal $Y_T$ and a suitable prefactor $c$. Assuming positively correlated $X$ and $Y_T$, $\kappa_1(t)$ is chosen to be a non-anticipating decaying exponential $\exp(-t/\tau_0)\Theta(t)$ with a time constant $\tau_0$ (in simulations $\tau_0 = 100$ ms), where $\Theta(t)$ is the Heaviside step function. This choice is motivated by the standard models for the impact of neuromodulators (see [7]), thus such a kernel may be implemented in a realistic biological mechanism. It turned out that the choice of $\tau_0$ was not critical, it could be varied over a decade ranging from 10 ms to 100 ms. The prefactor $c$ appearing in (7) can be determined from the fact that $F_1$ is the optimal linear estimator of the form given in (7), leading to:

$$c = \frac{\langle u_T(t), u(t) \rangle}{\langle u_T(t), u_T(t) \rangle}.$$

The quantity $c$ can be estimated online in the following way:

$$\frac{d}{dt} c(t) = (u_T(t) - \overline{u}_T(t)) \left[ (u(t) - \overline{u}(t)) - c(t)(u_T(t) - \overline{u}_T(t)) \right].$$

Using the above definitions, the resulting learning rule is given by (in vector notation):

$$\frac{d}{dt} w(t) = \alpha \frac{Y(t)\nu(t)}{u(t)\overline{u}(t)} \left[ -(u(t) - \overline{u}(t)) + c(t)\beta(u_T(t) - \overline{u}_T(t)) \right] - \alpha \lambda w(t). \quad (8)$$

Equation (8) will be called the spike-based learning rule, as the postsynaptic spike train $Y(t)$ explicitly appears. An accompanying rate-base learning rule can also be derived:

$$\frac{d}{dt} w(t) = \alpha \frac{\nu(t)}{u_0 \overline{u}(t)} \left[ -(u(t) - \overline{u}(t)) + c(t)\beta(u_T(t) - \overline{u}_T(t)) \right] - \alpha \lambda w(t). \quad (9)$$

# 3 Analytical results

The learning rules (8) and (9) are stochastic differential equations for the weights $w_j$ driven by the processes $Y(.)$, $\nu_j(.)$ and $u_T(.)$, of which the last two are assumed to be stationary with the means $\langle \nu_j(t) \rangle = \nu_0$ and $\langle u_T(t) \rangle = u_{T,0}$ respectively. The evolution of the solutions $w(t)$ to (8) and (9) may be studied via a Master equation for the probability distribution of the weights $p(w,t)$ (see [8]). For small learning rates $\alpha$, the stationary distribution $p(w)$ sharply peaks[3] at the roots of the drift function $A(w)$ of the corresponding Fokker-Planck equation (the detailed derivation is given in [4]). Thus, for $\alpha \ll 1$, the temporal evolution of the learning rules (8) and (9) may be studied via the deterministic differential equation:

$$\frac{d}{dt}\hat{w} \quad = \quad A(\hat{w}) = \alpha \frac{1}{\nu_0 u_0 z} \left( -C^0 + \beta C^1 \right) \hat{w} - \alpha \lambda \hat{w} \tag{10}$$

$$z \quad = \quad \sum_{j=1}^{N} \hat{w}_j, \tag{11}$$

where $z$ is the total weight. The matrix $C = -C^0 + \beta C^1$ (with the elements $C_{ij}$) has two contributions. $C^0$ is the covariance matrix of the input and the matrix $C^1$ quantifies the covariance between the activities $\nu_j$ and the trace $u_T$:

$$C_{ij}^0 \quad = \quad \langle \nu_i(t), \nu_j(t) \rangle$$

$$C_{ij}^1 \quad = \quad \frac{\langle \nu_i(t), u_T(t) \rangle \langle u_T(t), \nu_j(t) \rangle}{\langle u_T(t), u_T(t) \rangle}.$$

Now the critical points $w^*$ of dynamics of (10) are investigated. These critical points, if asymptotically stable, determine the peaks of the stationary distribution $p(w)$ of the weights $w$; we therefore expect the solutions of the stochastic equations to fluctuate around these fixed points $w^*$. If $\beta$ and $\lambda$ are much larger than one, the term containing the matrix $C^0$ can be neglected and equation (10) has a unique stable fixed point $w^*$:

$$w^* \quad \propto \quad C^T$$

$$C_i^T \quad = \quad \langle \nu_i(t), u_T(t) \rangle.$$

Under this assumption the maximal mutual information between the target signal $Y_T(t)$ and the output of the neuron $Y(t)$ is obtained by a weight vector $w = w^*$ that is parallel to the covariance vector $C^T$.

In general, the critical points of equation (10) depend on the eigenvalue spectrum of the symmetric matrix $C$: If all eigenvalues are negative, the weight vector $\hat{w}$ decays to the lower hard bound $0$. In case of at least one positive eigenvalue (which exists if $\beta$ is chosen large enough), there is a unique stable fixed point $w^*$:

$$w^* \quad = \quad \frac{\mu}{\lambda u_0 \nu_0 \bar{b}} b \tag{12}$$

$$\bar{b} \quad := \quad \sum_{i=1}^{N} b_i.$$

The vector $b$ appearing in (12) is the eigenvector of $C$ corresponding to the largest eigenvalue $\mu$. Thus, a stationary unimodal[4] distribution $p(w)$ of the weights $w$ is predicted, which is centered around the value $w^*$.

# 4 A concrete example for IB optimization

A special scenario of interest, that often appears in the literature (see for example [1], [9] and [10]), is the following: The synapses, and subsequently the input spike trains, form $M$ different subgroups

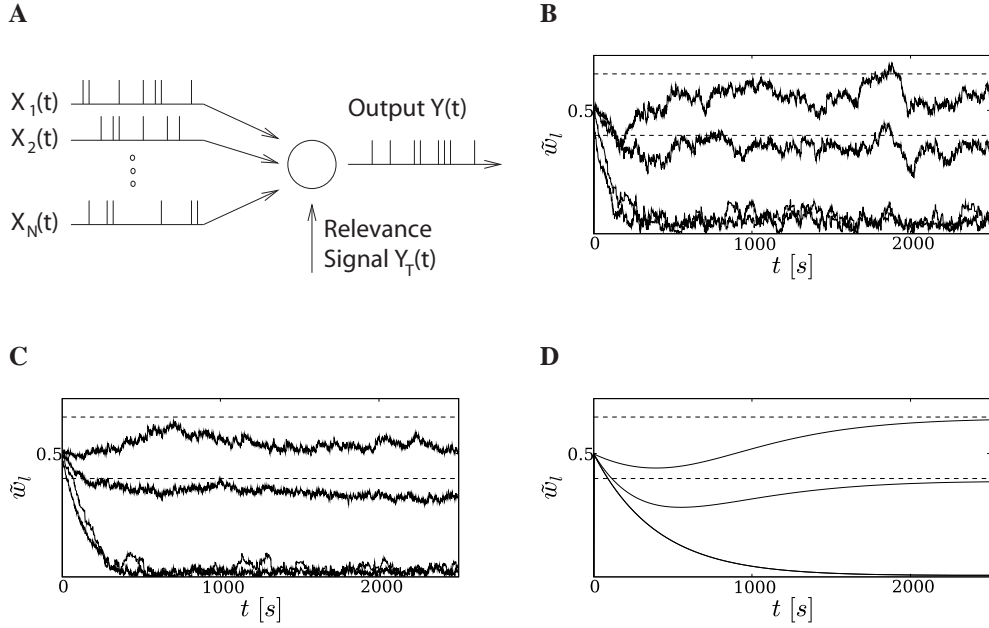

Figure 1: **A** The basic setup for the Information Bottleneck optimization. **B-D** Numerical and analytical results for the IB optimization task described in section 4. The temporal evolution of the average weights $\tilde{w}_l = 1/M \sum_{j \in G_l} w_j$ of the four different synaptic subgroups $G_l$ are shown. **B** The performance of the spike-based rule (8). The highest trajectory corresponds to $\tilde{w}_1$; it stays close to its analytical predicted fixed point value obtained from (12), which is visualized by the upper dashed line. The trajectory just below belongs to $\tilde{w}_3$, for which the fixed point value is also plotted as dashed line. The other two trajectories $\tilde{w}_2$ and $\tilde{w}_4$ decay and eventually fluctuate above the predicted value of zero. **C** The performance of the rate-based rule (9); results are analogous to the ones of the spike-based rule. **D** Simulation of the deterministic equation (10).

$G_l$, $l \in \{1, \ldots, N/M\}$ of the same size $N/M \in \mathbb{N}$. The spike trains $X_j$ and $X_k$, $j \neq k$, are statistically independent if they belong to different subgroups; within a subgroup there is a homogeneous covariance term $C^0_{jk} = c_l$, $j \neq k$ for $j, k \in G_l$, which can be due either to spike-spike correlations or correlations in rate modulations. The covariance between the target signal $Y_T$ and the spike trains $X_j$ is homogeneous among a subgroup.

As a numerical example, we consider in figure 1 a modification of the IB task presented in figure 2 of [1]. The $N = 100$ synapses form $M = 4$ subgroups $G_l = \{25(l-1)+1, \ldots, 25l\}$, $l \in \{1, \ldots, 4\}$. Synapses in $G_1$ receive Poisson spike trains of constant rate $\nu_0 = 20$ Hz, which are mutually spike-spike correlated with a correlation-coefficient[5] of 0.5. The same holds for the spike trains of $G_2$. Spike trains for $G_3$ and $G_4$ are uncorrelated Poisson trains with a common rate modulation, which is equal to low pass filtered white noise (cut-off frequency 5 Hz) with mean $\nu_0$ and standard deviation (SD) $\sigma = \nu_0/2$. The rate modulations for $G_3$ and $G_4$ are however independent (though identically distributed). Two spike trains for different synapse subgroups are statistically independent. The target signal $Y_T$ was chosen to be the sum of two Poisson trains. The first is of constant rate $\nu_0$ and has spike-spike correlations with $G_1$ of coefficient 0.5; the second is a Poisson spike train with the same rate modulation as the spike trains of $G_3$ superimposed by additional white noise of SD 2 Hz. Furthermore, the target signal was turned off during random intervals[6]. The resulting evolution of the weights is shown in figure 1, illustrating the performance of the spike-based rule (8) as well as of the rate-based rule (9). As expected, the weights of $G_1$ and $G_3$ are potentiated as $Y_T$ has mutual information with the corresponding part of the input. The synapses of $G_2$ and $G_4$ are depressed. The analytical result for the stable fixed point $w^*$ obtained from (12) is shown as dashed lines and is in good agreement with the numerical results. Furthermore the trajectory of the solution $\hat{w}(t)$ to

the deterministic equation (10) is plotted.

The presented concrete IB task was slightly changed from the one presented in [1], because for the setting used here, the largest eigenvalue $\mu$ of $C$ and its corresponding eigenvector $b$ can be calculated analytically. The simulation results for the original setting in [1] can also be reproduced with the simpler rules (8) and (9) (not shown).

## 5    Relevance-modulated PCA with spiking neurons

The presented learning rules (8) and (9) exhibit a close relation to Principal Component Analysis (PCA). A learning rule which enables the linear Poisson neuron to extract principal components from the input $X(.)$ can be derived by maximizing the following objective function:

$$L_{\mathrm{PCA}} = -L_{\mathrm{IB}} - \lambda \sum_{j=1}^{N} w_j^2 = +I(X,Y) - \beta I(Y_T,Y) - \lambda \sum_{j=1}^{N} w_j^2, \qquad (13)$$

which just differs from (3) by a change of sign in front of $L_{\mathrm{IB}}$. The resulting learning rule is in close analogy to (8):

$$\frac{d}{dt}w(t) = \alpha \frac{Y(t)\nu(t)}{u(t)\overline{u}(t)} \left[ (u(t) - \overline{u}(t)) - c(t)\beta(u_T(t) - \overline{u}_T(t)) \right] - \alpha\lambda w(t). \qquad (14)$$

The corresponding rate-based version can also be derived. Without the trace $u_T(.)$ of the target signal, it can be seen that the solution $\hat{w}(t)$ of deterministic equation corresponding to (14) (which is of the same form as (10) with the obvious sign changes) converges to an eigenvector of the covariance matrix $C^0$. Thus, for $\beta = 0$ we expect the learning rule (14) to perform PCA for small learning rates $\alpha$. The rule (14) without the relevance signal is comparable to other PCA rules, e. g. the covariance rule (see [11]) for non-spiking neurons.

The side information given by the relevance signal $Y_T(.)$ can be used to extract specific principal components from the input, thus we call this paradigm relevance-modulated PCA. Before we consider a concrete example for relevance-modulated PCA, we want to point out a further application of the learning rule (14).

The target signal $Y_T$ can also be used to extract different components from the input with different neurons (see figure 2). Consider $m$ neurons receiving the same input $X$. These neurons have the outputs $Y_1(.), \ldots, Y_m(t)$, target signals $Y_T^1(.), \ldots, Y_T^m(t)$ and weight vectors $w^1(t), \ldots, w^m(t)$, the latter evolving according to (14). In order to prevent all weight vectors from converging towards the same eigenvector of $C^0$ (the principal component), the target signal $Y_T^i$ for neuron $i$ is chosen to be the sum of all output spike trains except $Y_i$:

$$Y_T^i(t) = \sum_{j=1,\ j\neq i}^{N} Y_j(t). \qquad (15)$$

If one weight vector $w^i(t)$ is already close to the eigenvector $e^k$ of $C^0$, than by means of (15), the basins of attraction of $e^k$ for the other weight vectors $w^j(t)$, $j \neq i$ are reduced (or even vanish, depending on the value of $\beta$). It is therefore less likely (or impossible) that they also converge to $e^k$. In practice, this setup is sufficiently robust, if only a small number ($\leq 4$) of different components is to be extracted and if the differences between the eigenvalues $\lambda_i$ of these principal components are not too big[7]. For the PCA learning rule, the time constant $\tau_0$ of the kernel $\kappa_1$ (see (7)) had to be chosen smaller than for the IB tasks in order to obtain good performance; we used $\tau_0 = 10\,\mathrm{ms}$ in simulations. This is in the range of time constants for IPSPs. Hence, the signals $Y_T^i$ could probably be implemented via lateral inhibition.

The learning rule considered in [3] displayed a close relation to Independent Component Analysis (ICA). Because of the linear neuron model used here and the linearization of further terms in the derivation, the resulting learning rule (14) performs PCA instead of ICA.

The results of a numerical example are shown in figure 2. The $m = 3$ for the regular PCA experiment neurons receive the same input $X$ and their weights change according to (14). The weights and input spike trains are grouped into four subgroups $G_1, \ldots, G_4$, as for the IB optimization discussed

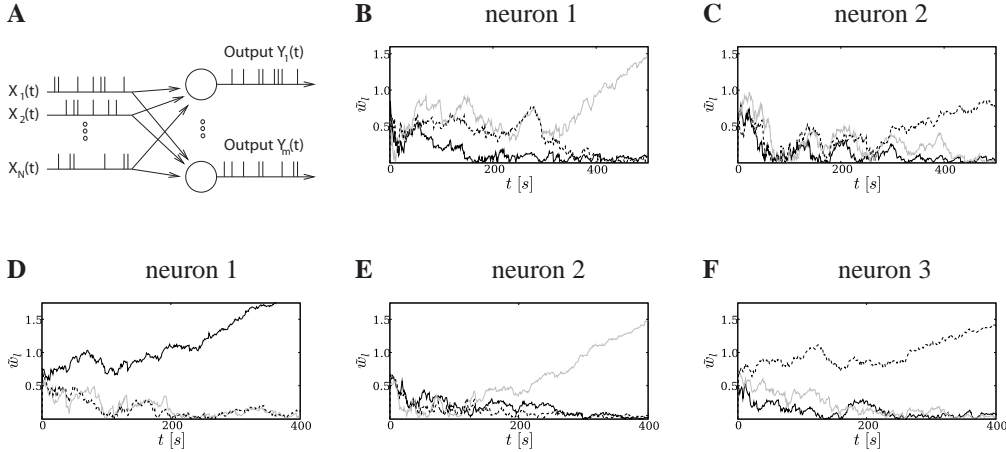

Figure 2: **A** The basic setup for the PCA task: The $m$ different neurons receive the same input $X$ and are expected to extract different principal components of it. **B-F** The temporal evolution of the average subgroup weights $\tilde{w}_l = 1/25 \sum_{j \in G_l} w_j$ for the groups $G_1$ (black solid line), $G_2$ (light gray solid line) and $G_3$ (dotted line). **B-C** Results for the relevance-modulated PCA task: neuron 1 (fig. **B**) specializes on $G_2$ and neuron 2 (fig. **C**) on subgroup $G_3$. **D-F** Results for the regular PCA task: neuron 1 (fig. **D**) specialize on $G_1$, neuron 2 (fig. **E**) on $G_2$ and neuron 3 (fig. **F**) on $G_3$ .

in section 4. The only difference is that all groups (except for $G_4$) receive spike-spike correlated Poisson spike trains with a correlation coefficient for the groups $G_1$, $G_2$, $G_3$ of 0.5, 0.45, 0.4 respectively. Group $G_4$ receives uncorrelated Poisson spike trains. As can be seen in figure 2 **D** to **F**, the different neurons specialize on different principal components corresponding to potentiated synaptic subgroups $G_1$, $G_2$ and $G_3$ respectively. Without the relevance signals $Y_T^i(.)$, all neurons tend to specialize on the principal component corresponding to $G_1$ (not shown).

As a concrete example for relevance-modulated PCA, we consider the above setup with slight modifications: Now we want $m = 2$ neurons to extract the components $G_2$ and $G_3$ from the input $X$, and not the principal component $G_1$. This is achieved with an additional relevance signal $Y_T^0$, which is the same for both neurons and has spike-spike correlations with $G_2$ and $G_3$ of 0.45 and 0.4. We add the term $\gamma I(Y, Y_T^0)$ to the objective function (13), where $\gamma$ is a positive trade-off factor. The resulting learning rule has exactly the same structure as (14), with an additional term due to $Y_T^0$. The numerical results are presented in figure 2 **B** and **C**, showing that it is possible in this setup to explicitly select the principle components that are extracted (or not extracted) by the neurons.

## 6 Discussion

We have introduced and analyzed a simple and perspicuous rule that enables spiking neurons to perform IB optimization in an online manner. Our simulations show that this rule works as well as the substantially more complex learning rule that had previously been proposed in [3]. It also performs well for more realistic neuron models as indicated in [4]. We have shown that the convergence properties of our simplified IB rule can be analyzed with the help of the Fokker-Planck equation (alternatively one may also use the theoretical framework described in A.2 in [12] for its analysis). The investigation of the weight vectors to which this rule converges reveals interesting relationships to PCA. Apparently, very little is known about learning rules that enable spiking neurons to extract multiple principal components from an input stream (a discussion of a basic learning rule performing PCA is given in chapter 11.2.4 of [5]). We have demonstrated both analytically and through simulations that a slight variation of our new learning rule performs PCA. Our derivation of this rule within the IB framework opens the door to new variations of PCA where preferentially those components are extracted from a high dimensional input stream that are –or are not– related to some external relevance variable. We expect that a further investigation of such methods will shed light on the unknown principles of unsupervised and semi-supervised learning that might shape and constantly retune the output of lower cortical areas to intermediate and higher cortical areas. The learning rule that we have proposed might in principle be able to extract from high-dimensional

sensory input streams $X$ those components that are related to other sensory modalities or to internal expectations and goals.

Quantitative biological data on the precise way in which relevance signals $Y_T$ (such as for example dopamin) might reach neurons in the cortex and modulate their synaptic plasticity are still missing. But it is fair to assume that these signals reach the synapse in a low-pass filtered form of the type $u_T$ that we have assumed for our learning rules. From that perspective one can view the learning rules that we have derived (in contrast to the rules proposed in [3]) as local learning rules.

**Acknowledgments**

Written under partial support by the Austrian Science Fund FWF, project # P17229, project # S9102 and project # FP6-015879 (FACETS) of the European Union.

## Footnotes

[1]The term $D_{KL}(P(Y)\|P(\tilde{Y}))$ denotes the Kullback-Leibler divergence between the distribution $P(Y)$ and a target distribution $P(\tilde{Y})$. This term ensures that the weights remain bounded, it is shortly discussed in [4].

[2]The remedy, proposed in section 3.1 in [3], of replacing the mutual information $I(Y, Y_T)$ in $L_0$ by an information rate $I(Y, Y_T)/\Delta t$ does not solve this problem, as the term $I(Y, Y_T)/\Delta t$ diverges in the continuous time limit.

[3]It can be shown that the diffusion term in the FP equation scales like $\mathcal{O}(\alpha)$, i. e. for small learning rates $\alpha$, fluctuations tend to zero and the dynamics can be approximated by the differential equation (10) .

[4]Note that $p(w)$ denotes the distribution of the weight vector, not the distribution of a single weight $p(w_j)$.

[5]Spike-spike correlated Poisson spike trains were generated according to the method outlined in [9].

[6]These intervals of silence were modeled as random telegraph noise with a time constant of 200 ms and a overall probability of silence of 0.5.

[7]Note that the input $X$ may well exhibit a much larger number of principal components. However it is only possible to extract a limited number of them by different neurons at the same time.

## References

[1] S. Klampfl, R. A. Legenstein, and W. Maass. Information bottleneck optimization and independent component extraction with spiking neurons. In *Proc. of NIPS 2006, Advances in Neural Information Processing Systems*, volume 19. MIT Press, 2007.

[2] N. Tishby, F. C. Pereira, and W. Bialek. The information bottleneck method. In *Proceedings of the 37-th Annual Allerton Conference on Communication, Control and Computing*, pages 368–377, 1999.

[3] S. Klampfl, R. Legenstein, and W. Maass. Spiking neurons can learn to solve information bottleneck problems and to extract independent components. *Neural Computation*, 2007. in press.

[4] L. Buesing and W. Maass. journal version. 2007. in preparation.

[5] W. Gerstner and W. M. Kistler. *Spiking Neuron Models*. Cambridge University Press, Cambridge, 2002.

[6] Taro Toyoizumi, Jean-Pascal Pfister, Kazuyuki Aihara, and Wulfram Gerstner. Optimality Model of Unsupervised Spike-Timing Dependent Plasticity: Synaptic Memory and Weight Distribution. *Neural Computation*, 19(3):639–671, 2007.

[7] Eugene M. Izhikevich. Solving the Distal Reward Problem through Linkage of STDP and Dopamine Signaling. *Cereb. Cortex*, page bhl152, 2007.

[8] H. Risken. *The Fokker-Planck Equation*. Springer, 3rd edition, 1996.

[9] R. Gütig, R. Aharonov, S. Rotter, and H. Sompolinsky. Learning input correlations through non-linear temporally asymmetric hebbian plasticity. *Journal of Neurosci.*, 23:3697–3714, 2003.

[10] H. Meffin, J. Besson, A. N. Burkitt, and D. B. Grayden. Learning the structure of correlated synaptic subgroups using stable and competitive spike-timing-dependent plasticity. *Physical Review E*, 73, 2006.

[11] T. J. Sejnowski and G. Tesauro. The hebb rule for synaptic plasticity: algorithms and implementations. In J. H. Byrne and W. O. Berry, editors, *Neural Models of Plasticity*, pages 94–103. Academic Press, 1989.

[12] N. Intrator and L. N. Cooper. Objective function formulation of the BCM theory of visual cortical plasticity: statistical connections, stability conditions. *Neural Networks*, 5:3–17, 1992.

